# Variational Gaussian-process factor analysis for modeling spatio-temporal data

**Jaakko Luttinen**
Adaptive Informatics Research Center
Helsinki University of Technology, Finland
Jaakko.Luttinen@tkk.fi

**Alexander Ilin**
Adaptive Informatics Research Center
Helsinki University of Technology, Finland
Alexander.Ilin@tkk.fi

## Abstract

We present a probabilistic factor analysis model which can be used for studying spatio-temporal datasets. The spatial and temporal structure is modeled by using Gaussian process priors both for the loading matrix and the factors. The posterior distributions are approximated using the variational Bayesian framework. High computational cost of Gaussian process modeling is reduced by using sparse approximations. The model is used to compute the reconstructions of the global sea surface temperatures from a historical dataset. The results suggest that the proposed model can outperform the state-of-the-art reconstruction systems.

## 1 Introduction

Factor analysis and principal component analysis (PCA) are widely used linear techniques for finding dominant patterns in multivariate datasets. These methods find the most prominent correlations in the data and therefore they facilitate studies of the observed system. The found principal patterns can also give an insight into the observed data variability. In many applications, the quality of this kind of modeling can be significantly improved if extra knowledge about the data structure is used. For example, taking into account the temporal information typically leads to more accurate modeling of time series.

In this work, we present a factor analysis model which makes use of both temporal and spatial information for a set of collected data. The method is based on the standard factor analysis model

$$\mathbf{Y} = \mathbf{W}\mathbf{X} + \text{noise} = \sum_{d=1}^{D} \mathbf{w}_{:d}\mathbf{x}_{d:}^{\mathrm{T}} + \text{noise}\,, \tag{1}$$

where $\mathbf{Y}$ is a matrix of spatio-temporal data in which each row contains measurements in one spatial location and each column corresponds to one time instance. Here and in the following, we denote by $\mathbf{a}_{i:}$ and $\mathbf{a}_{:i}$ the $i$-th row and column of a matrix $\mathbf{A}$, respectively (both are column vectors). Thus, each $\mathbf{x}_{d:}$ represents the time series of one of the $D$ factors whereas $\mathbf{w}_{:d}$ is a vector of loadings which are spatially distributed. The matrix $\mathbf{Y}$ can contain missing values and the samples can be unevenly distributed in space and time.[1]

We assume that both the factors $\mathbf{x}_{d:}$ and the corresponding loadings $\mathbf{w}_{:d}$ have prominent structures. We describe them by using Gaussian processes (GPs) which is a flexible and theoretically solid tool for smoothing and interpolating non-uniform data [8]. Using separate GP models for $\mathbf{x}_{d:}$ and $\mathbf{w}_{:d}$ facilitates analysis of large spatio-temporal datasets. The application of the GP methodology to modeling data $\mathbf{Y}$ directly could be unfeasible in real-world problems because the computational

complexity of inference scales cubically w.r.t. the number of data points. The advantage of the proposed approach is that we perform GP modeling only either in the spatial or temporal domain at a time. Thus, the dimensionality can be remarkably reduced and modeling large datasets becomes feasible. Also, good interpretability of the model makes it easy to explore the results in the spatial and temporal domain and to set priors reflecting our modeling assumptions. The proposed model is symmetrical w.r.t. space and time.

Our model bears similarities with the latent variable models presented in [13, 16]. There, GPs were used to describe the factors and the mixing matrix was point-estimated. Therefore the observations $\mathbf{Y}$ were modeled with a GP. In contrast to that, our model is not a GP model for the observations because the marginal distribution of $\mathbf{Y}$ is not Gaussian. This makes the posterior distribution of the unknown parameters intractable. Therefore we use an approximation based on the variational Bayesian methodology. We also show how to use sparse variational approximations to reduce the computational load. Models which use GP priors for both $\mathbf{W}$ and $\mathbf{X}$ in (1) have recently been proposed in [10, 11]. The function factorization model in [10] is learned using a Markov chain Monte Carlo sampling procedure, which may be computationally infeasible for large-scale datasets. The nonnegative matrix factorization model in [11] uses point-estimates for the unknown parameters, thus ignoring posterior uncertainties. In our method, we take into account posterior uncertainties, which helps reduce overfitting and facilitates learning a more accurate model.

In the experimental part, we use the model to compute reconstruction of missing values in a real-world spatio-temporal dataset. We use a historical sea surface temperature dataset which contains monthly anomalies in the 1856-1991 period to reconstruct the global sea surface temperatures. The same dataset was used in designing the state-of-the-art reconstruction methodology [5]. We show the advantages of the proposed method as a Bayesian technique which can incorporate all assumptions in one model and which uses all available data. Since reconstruction of missing values can be an important application for the method, we give all the formulas assuming missing values in the data matrix $\mathbf{Y}$.

## 2   Factor analysis model with Gaussian process priors

We use the factor analysis model (1) in which $\mathbf{Y}$ has dimensionality $M \times N$ and the number of factors $D$ is much smaller than the number of spatial locations $M$ and the number of time instances $N$. The $m$-th row of $\mathbf{Y}$ corresponds to a spatial location $l_m$ (e.g., a location on a two-dimensional map) and the $n$-th column corresponds to a time instance $t_n$.

We assume that each time signal $\mathbf{x}_{d:}$ contains values of a latent function $\chi(t)$ computed at time instances $t_n$. We use independent Gaussian process priors to describe each signal $\mathbf{x}_{d:}$:

$$p(\mathbf{X}) = \mathcal{N}\left(\mathbf{X}_{:} \,|\, 0, \mathbf{K}_{\mathbf{x}}\right) = \prod_{d=1}^{D} \mathcal{N}\left(\mathbf{x}_{d:} \,|\, 0, \mathbf{K}_d\right), \qquad [\mathbf{K}_d]_{ij} = \psi_d(t_i, t_j; \boldsymbol{\theta}_d), \qquad (2)$$

where $\mathbf{X}_{:}$ denotes a long vector formed by concatenating the columns of $\mathbf{X}$, $\mathbf{K}_d$ is the part of the large covariance matrix $\mathbf{K}_{\mathbf{x}}$ which corresponds to the $d$-th row of $\mathbf{X}$ and $\mathcal{N}\left(\mathbf{a} \,|\, \mathbf{b}, \mathbf{C}\right)$ denotes the Gaussian probability density function for variable $\mathbf{a}$ with mean $\mathbf{b}$ and covariance $\mathbf{C}$. The $ij$-th element of $\mathbf{K}_d$ is computed using the covariance function $\psi_d$ with the kernel hyperparameters $\boldsymbol{\theta}_d$.

The priors for $\mathbf{W}$ are defined similarly assuming that each spatial pattern $\mathbf{w}_{:d}$ contains measurements of a function $\omega(l)$ at different spatial locations $l_m$:

$$p(\mathbf{W}) = \prod_{d=1}^{D} \mathcal{N}\left(\mathbf{w}_{:d} \,|\, 0, \mathbf{K}_d^{\mathbf{w}}\right), \qquad [\mathbf{K}_d^{\mathbf{w}}]_{ij} = \varphi_d(l_i, l_j; \boldsymbol{\phi}_d), \qquad (3)$$

where $\varphi_d$ is a covariance function with hyperparameters $\boldsymbol{\phi}_d$. Any valid (positive semidefinite) kernels can be used to define the covariance functions $\psi_d$ and $\varphi_d$. A good list of possible covariance functions is given in [8]. The prior model reduces to the one used in probabilistic PCA [14] when $\mathbf{K}_d = \mathbf{I}$ and a uniform prior is used for $\mathbf{W}$.

The noise term in (1) is modeled with a Gaussian distribution, resulting in a likelihood function

$$p(\mathbf{Y} | \mathbf{W}, \mathbf{X}, \boldsymbol{\sigma}) = \prod_{mn \in \mathcal{O}} \mathcal{N}\left(y_{mn} \,|\, \mathbf{w}_{m:}^{\mathrm{T}} \mathbf{x}_{:n}, \sigma_{mn}^2\right), \qquad (4)$$

where the product is evaluated over the observed elements in $\mathbf{Y}$ whose indices are included in the set $\mathcal{O}$. We will refer to the model (1)–(4) as GPFA. In practice, the noise level can be assumed spatially ($\sigma_{mn} = \sigma_m$) or temporally ($\sigma_{mn} = \sigma_n$) varying. One can also use spatially and temporally varying noise level $\sigma_{mn}^2$ if this variability can be estimated somehow.

There are two main difficulties which should be addressed when learning the model: 1) The posterior $p(\mathbf{W}, \mathbf{X}|\mathbf{Y})$ is intractable and 2) the computational load for dealing with GPs can be too large for real-world datasets. We use the variational Bayesian framework to cope with the first difficulty and we also adopt the variational approach when computing sparse approximations for the GP posterior.

# 3   Learning algorithm

In the variational Bayesian framework, the true posterior is approximated using some restricted class of possible distributions. An approximate distribution which factorizes as

$$p(\mathbf{W}, \mathbf{X}|\mathbf{Y}) \approx q(\mathbf{W}, \mathbf{X}) = q(\mathbf{W})q(\mathbf{X}) \, .$$

is typically used for factor analysis models. The approximation $q(\mathbf{W}, \mathbf{X})$ can be found by minimizing the Kullback-Leibler divergence from the true posterior. This optimization is equivalent to the maximization of the lower bound of the marginal log-likelihood:

$$\log p(\mathbf{Y}) \geq \int q(\mathbf{W})q(\mathbf{X}) \log \frac{p(\mathbf{Y}|\mathbf{W}, \mathbf{X})p(\mathbf{W})p(\mathbf{X})}{q(\mathbf{W})q(\mathbf{X})} d\mathbf{W} d\mathbf{X} \, . \tag{5}$$

Free-form maximization of (5) w.r.t. $q(\mathbf{X})$ yields that

$$q(\mathbf{X}) \propto p(\mathbf{X}) \exp^{\langle \log p(\mathbf{Y}|\mathbf{W}, \mathbf{X}) \rangle_{q(\mathbf{W})}} \, ,$$

where $\langle \cdot \rangle$ refers to the expectation over the approximate posterior distribution $q$. Omitting the derivations here, this boils down to the following update rule:

$$q(\mathbf{X}) = \mathcal{N} \left( \mathbf{X}_{:} \left| \left( \mathbf{K}_{\mathbf{x}}^{-1} + \mathbf{U} \right)^{-1} \mathbf{Z}_{:}, \left( \mathbf{K}_{\mathbf{x}}^{-1} + \mathbf{U} \right)^{-1} \right) \right. , \tag{6}$$

where $\mathbf{Z}_{:}$ is a $DN \times 1$ vector formed by concatenation of vectors

$$\mathbf{z}_{:n} = \sum_{m \in \mathcal{O}_n} \sigma_{mn}^{-2} \langle \mathbf{w}_{m:} \rangle y_{mn} \, . \tag{7}$$

The summation in (7) is over a set $\mathcal{O}_n$ of indices $m$ for which $y_{mn}$ is observed. Matrix $\mathbf{U}$ in (6) is a $DN \times DN$ block-diagonal matrix with the following $D \times D$ matrices on the diagonal:

$$\mathbf{U}_n = \sum_{m \in \mathcal{O}_n} \sigma_{mn}^{-2} \langle \mathbf{w}_{m:} \mathbf{w}_{m:}^{\mathrm{T}} \rangle \, , \qquad n = 1, \ldots, N \, . \tag{8}$$

Note that the form of the approximate posterior (6) is similar to the regular GP regression: One can interpret $\mathbf{U}_n^{-1} \mathbf{z}_{:n}$ as noisy observations with the corresponding noise covariance matrices $\mathbf{U}_n^{-1}$. Then, $q(\mathbf{X})$ in (6) is simply the posterior distribution of the latent functions values $\chi_d(t_n)$.

The optimal $q(\mathbf{W})$ can be computed using formulas symmetrical to (6)–(8) in which $\mathbf{X}$ and $\mathbf{W}$ are appropriately exchanged. The variational EM algorithm for learning the model consists of alternate updates of $q(\mathbf{W})$ and $q(\mathbf{X})$ until convergence. The noise level can be estimated by using a point estimate or adding a factor factor $q(\sigma_{mn})$ to the approximate posterior distribution. For example, the update rules for the case of isotropic noise $\sigma_{mn}^2 = \sigma^2$ are given in [2].

## 3.1   Component-wise factorization

In practice, one may need to factorize further the posterior approximation in order to reduce the computational burden. This can be done in two ways: by neglecting the posterior correlations between different factors $\mathbf{x}_{d:}$ (and between spatial patterns $\mathbf{w}_{:d}$, respectively) or by neglecting the posterior correlations between different time instances $\mathbf{x}_{:n}$ (and between spatial locations $\mathbf{w}_{m:}$, respectively). We suggest to use the first way which is computationally more expensive but allows to

| Method | Approximation | Update rule | Complexity |
|---|---|---|---|
| GP on $\mathbf{Y}$ | | | $O(N^3 M^3)$ |
| GPFA | $q(\mathbf{X}_:)$ | (6) | $O(D^3 N^3 + D^3 M^3)$ |
| GPFA | $q(\mathbf{x}_{d:})$ | (9) | $O(DN^3 + DM^3)$ |
| GPFA | $q(\mathbf{x}_{d:})$, inducing inputs | (12) | $O(\sum_{d=1}^{D} N_d^2 N + \sum_{d=1}^{D} M_d^2 M)$ |

Table 1: The computational complexity of different algorithms

capture stronger posterior correlations. This yields a posterior approximation $q(\mathbf{X}) = \prod_{d=1}^{D} q(\mathbf{x}_{d:})$ which can be updated as follows:

$$q(\mathbf{x}_{d:}) = \mathcal{N}\left(\mathbf{x}_{d:} \left| \left(\mathbf{K}_d^{-1} + \mathbf{V}_d\right)^{-1} \mathbf{c}_d, \left(\mathbf{K}_d^{-1} + \mathbf{V}_d\right)^{-1}\right.\right), \qquad d = 1, \ldots, D, \qquad (9)$$

where $\mathbf{c}_d$ is an $N \times 1$ vector whose $n$-th component is

$$[\mathbf{c}_d]_n = \sum_{m \in \mathcal{O}_n} \sigma_{mn}^{-2} \langle w_{md} \rangle \left( y_{mn} - \sum_{j \neq d} \langle w_{mj} \rangle \langle x_{jn} \rangle \right) \qquad (10)$$

and $\boldsymbol{V}_d$ is an $N \times N$ diagonal matrix whose $n$-th diagonal element is $[\mathbf{V}_d]_{nn} = \sum_{m \in \mathcal{O}_n} \sigma_{mn}^{-2} \langle w_{md}^2 \rangle$. The main difference to (6) is that each component is fitted to the residuals of the reconstruction based on the rest of the components. The computational complexity is now reduced compared to (9), as shown in Table 1.

The component-wise factorization may provide a meaningful representation of data because the model is biased in favor of solutions with dynamically and spatially decoupled components. When the factors are modeled using rather general covariance functions, the proposed method is somewhat related to the blind source separation techniques using time structure (e.g., [1]). The advantage here is that the method can handle more sophisticated temporal correlations and it is easily applicable to incomplete data. In addition, one can use the method in semi-blind settings when prior knowledge is used to extract components with specific types of temporal or spatial features [9]. This problem can be addressed using the proposed technique with properly chosen covariance functions.

## 3.2 Variational learning of sparse GP approximations

One of the main issues with Gaussian processes is the high computational cost with respect to the number of observations. Although the variational learning of the GPFA model works only in either spatial or temporal domain at a time, the size of the data may still be too large in practice. A common way to reduce the computational cost is to use sparse approximations [7]. In this work, we follow the variational formulation of sparse approximations presented in [15].

The main idea is to introduce a set of auxiliary variables $\{\boldsymbol{w}, \boldsymbol{x}\}$ which contain the values of the latent functions $\omega_d(l)$, $\chi_d(t)$ in some locations $\{l = \lambda_m^d | m = 1, \ldots, M_d\}$, $\{t = \tau_n^d | n = 1, \ldots, N_d\}$ called inducing inputs. Assuming that the auxiliary variables $\{\boldsymbol{w}, \boldsymbol{x}\}$ summarize the data well, it holds that $p(\mathbf{W}, \mathbf{X} | \boldsymbol{w}, \boldsymbol{x}, \mathbf{Y}) \approx p(\mathbf{W}, \mathbf{X} | \boldsymbol{w}, \boldsymbol{x})$, which suggests a convenient form of the approximate posterior:

$$q(\mathbf{W}, \mathbf{X}, \boldsymbol{w}, \boldsymbol{x}) = p(\mathbf{W} | \boldsymbol{w}) p(\mathbf{X} | \boldsymbol{x}) q(\boldsymbol{w}) q(\boldsymbol{x}), \qquad (11)$$

where $p(\mathbf{W} | \boldsymbol{w})$, $p(\mathbf{X} | \boldsymbol{x})$ can be easily computed from the GP priors. Optimal $q(\boldsymbol{w})$, $q(\boldsymbol{x})$ can be computed by maximizing the variational lower bound of the marginal log-likelihood similar to (5).

Free-form maximization w.r.t. $q(\boldsymbol{x})$ yields the following update rule:

$$q(\boldsymbol{x}) = \mathcal{N}\left(\boldsymbol{x} \left| \boldsymbol{\Sigma} \mathbf{K}_{\boldsymbol{x}}^{-1} \mathbf{K}_{\boldsymbol{x}\mathbf{x}} \mathbf{Z}_:, \boldsymbol{\Sigma}\right.\right), \quad \boldsymbol{\Sigma} = \left(\mathbf{K}_{\boldsymbol{x}}^{-1} + \mathbf{K}_{\boldsymbol{x}}^{-1} \mathbf{K}_{\boldsymbol{x}\mathbf{x}} \mathbf{U} \mathbf{K}_{\mathbf{x}\boldsymbol{x}} \mathbf{K}_{\boldsymbol{x}}^{-1}\right)^{-1}, \qquad (12)$$

where $\boldsymbol{x}$ is the vector of concatenated auxiliary variables for all factors, $\mathbf{K}_{\boldsymbol{x}}$ is the GP prior covariance matrix of $\boldsymbol{x}$ and $\mathbf{K}_{\boldsymbol{x}\mathbf{x}}$ is the covariance between $\boldsymbol{x}$ and $\mathbf{X}_:$. This equation can be seen as a replacement of (6). A similar formula is applicable to the update of $q(\boldsymbol{w})$. The advantage here is that the number of inducing inputs is smaller than then the number of data samples, that is, $M_d < M$ and $N_d < N$, and therefore the required computational load can be reduced (see more details in [15]). Eq. (12) can be quite easily adapted to the component-wise factorization of the posterior in order to reduce the computational load of (9). See the summary for the computational complexity in Table 1.

### 3.3 Update of GP hyperparameters

The hyperparameters of the GP priors can be updated quite similarly to the standard GP regression by maximizing the lower bound of the marginal log-likelihood. Omitting the derivations here, this lower bound for the temporal covariance functions $\{\psi_d(t)\}_{d=1}^D$ equals (up to a constant) to

$$\log \mathcal{N} \left( \mathbf{U}^{-1}\mathbf{Z}_: \,\middle|\, 0, \mathbf{U}^{-1} + \mathbf{K_{xx}}\mathbf{K}_{\boldsymbol{x}}^{-1}\mathbf{K}_{\boldsymbol{x}\mathbf{x}} \right) - \frac{1}{2}\,\mathrm{tr}\left[ \sum_{n=1}^N \mathbf{U}_n\mathbf{D} \right], \tag{13}$$

where $\mathbf{U}$ and $\mathbf{Z}_:$ have the same meaning as in (6) and $\mathbf{D}$ is a $D \times D$ (diagonal) matrix of variances of $\mathbf{x}_{:n}$ given the auxiliary variables $\boldsymbol{x}$. The required gradients are shown in the appendix. The equations without the use of auxiliary variables are similar except that $\mathbf{K_{xx}}\mathbf{K}_{\boldsymbol{x}}^{-1}\mathbf{K}_{\boldsymbol{x}\mathbf{x}} = \mathbf{K_x}$ and the second term disappears. A symmetrical equation can be derived for the hyperparameters of the spatial functions $\varphi_d(t)$. The extension of (13) to the case of component-wise factorial approximation is straightforward. The inducing inputs can also be treated as variational parameters and they can be changed to optimize the lower bound (13).

## 4 Experiments

### 4.1 Artificial example

We generated a dataset with $M = 30$ sensors (two-dimensional spatial locations) and $N = 200$ time instances using the generative model (1) with a moderate amount of observation noise, assuming $\sigma_{mn} = \sigma$. $D = 4$ temporal signals $\mathbf{x}_{d:}$ were generated by taking samples from GP priors with different covariance functions: 1) a squared exponential function to model a slowly changing component:

$$k(r; \theta_1) = \exp\left(-\frac{r^2}{2\theta_1^2}\right), \tag{14}$$

2) a periodic function with decay to model a quasi-periodic component:

$$k(r; \theta_1, \theta_2, \theta_3) = \exp\left(-\frac{2\sin^2(\pi r/\theta_1)}{\theta_2^2} - \frac{r^2}{2\theta_3^2}\right), \tag{15}$$

where $r = |t_j - t_i|$, and 3) a compactly supported piecewise polynomial function to model two fast changing components with different timescales:

$$k(r; \theta_1) = \frac{1}{3}(1 - r)^{b+2}\left((b^2 + 4b + 3)r^2 + (3b + 6)r + 3\right), \tag{16}$$

where $r = \min(1, |t_j - t_i|/\theta_1)$ and $b = 3$ for one-dimensional inputs with the hyperparameter $\theta_1$ defining a threshold such that $k(r) = 0$ for $|t_j - t_i| \geq \theta_1$. The loadings were generated from GPs over the two-dimensional space using the squared exponential covariance function (14) with an additional scale parameter $\theta_2$:

$$k(r; \theta_1, \theta_2) = \theta_2^2 \exp\left(-r^2/(2\theta_1^2)\right). \tag{17}$$

We randomly selected 452 data points from $\mathbf{Y}$ as being observed, thus most of the generated data points were marked as missing (see Fig. 1a for examples). We also removed observations from all the sensors for a relatively long time interval. Note a resulting gap in the data marked with vertical lines in Fig. 1a. The hyperparameters of the Gaussian processes were initialized randomly close to the values used for data generation, assuming that a good guess about the hidden signals can be obtained by exploratory analysis of data.

Fig. 1b shows the components recovered by GPFA using the update rule (6). Note that the algorithm separated the four signals with the different variability timescales. The posterior predictive distributions of the missing values presented in Fig. 1a show that the method was able to capture temporal correlations on different timescales. Note also that although some of the sensors contain very few observations, the missing values are reconstructed pretty well. This is a positive effect of the spatially smooth priors.

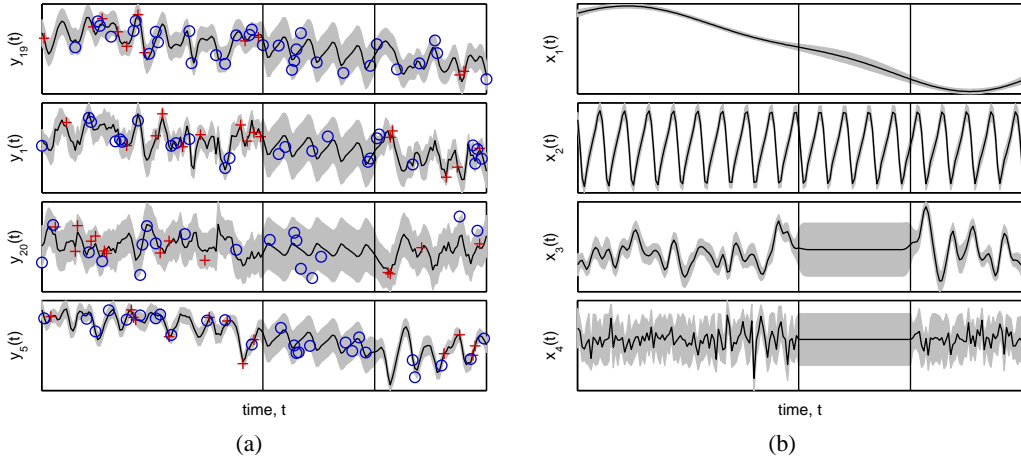

Figure 1: Results for the artificial experiment. (a) Posterior predictive distribution for four randomly selected locations with the observations shown as crosses, the gap with no training observations marked with vertical lines and some test values shown as circles. (b) The posteriors of the four latent signals $\mathbf{x}_{d:}$. In both figures, the solid lines show the posterior mean and gray color shows two standard deviations.

## 4.2 Reconstruction of global SST using the MOHSST5 dataset

We demonstrate how the presented model can be used to reconstruct global sea surface temperatures (SST) from historical measurements. We use the U.K. Meteorological Office historical SST data set (MOHSST5) [6] that contain monthly SST anomalies in the 1856-1991 period for $5° \times 5°$ longitude-latitude bins. The dataset contains in total approximately 1600 time instances and 1700 spatial locations. The dataset is sparse, especially during the 19th century and the World Wars, having $55\%$ of the values missing, and thus, consisting of more than $10^6$ observations in total.

We used the proposed algorithm to estimate $D = 80$ components, the same number was used in [5]. We withdrew 20% of the data from the training set and used this part for testing the reconstruction accuracy. We used five time signals $\mathbf{x}_{d:}$ with the squared exponential function (14) to describe climate trends. Another five temporal components were modeled with the quasi-periodic covariance function (15) to capture periodic signals (e.g. related to the annual cycle). We also used five components with the squared exponential function to model prominent interannual phenomena such as El Niño. Finally we used the piecewise polynomial functions to describe the rest 65 time signals $\mathbf{x}_{d:}$. These dimensionalities were chosen ad hoc. The covariance function for each spatial pattern $\mathbf{w}_{:d}$ was the scaled squared exponential (17). The distance $r$ between the locations $l_i$ and $l_j$ was measured on the surface of the Earth using the spherical law of cosines. The use of the extra parameter $\theta_2$ in (17) allowed automatic pruning of unnecessary factors, which happens when $\theta_2 = 0$.

We used the component-wise factorial approximation of the posterior described in Section 3.1. We also introduced 500 inducing inputs for each spatial function $\omega_d(l)$ in order to use sparse variational approximations. Similar sparse approximations were used for the 15 temporal functions $\chi(t)$ which modeled slow climate variability: the slowest, quasi-periodic and interannual components had 80, 300 and 300 inducing inputs, respectively. The inducing inputs were initialized by taking a random subset from the original inputs and then kept fixed throughout learning because their optimization would have increased the computational burden substantially. For the rest of the temporal phenomena, we used the piecewise polynomial functions (16) that produce priors with a sparse covariance matrix and therefore allow efficient computations.

The dataset was preprocessed by weighting the data points by the square root of the corresponding latitudes in order to diminish the effect of denser sampling in the polar regions, then the same noise level was assumed for all measurements ($\sigma_{mn} = \sigma$). Preprocessing by weighting data points $y_{mn}$ with weights $s_m$ is essentially equivalent to assuming spatially varying noise level $\sigma_{mn} = \sigma/s_m$. The GP hyperparameters were initialized taking into account the assumed smoothness of the spatial patterns and the variability timescale of the temporal factors. The factors $\mathbf{X}$ were initialized

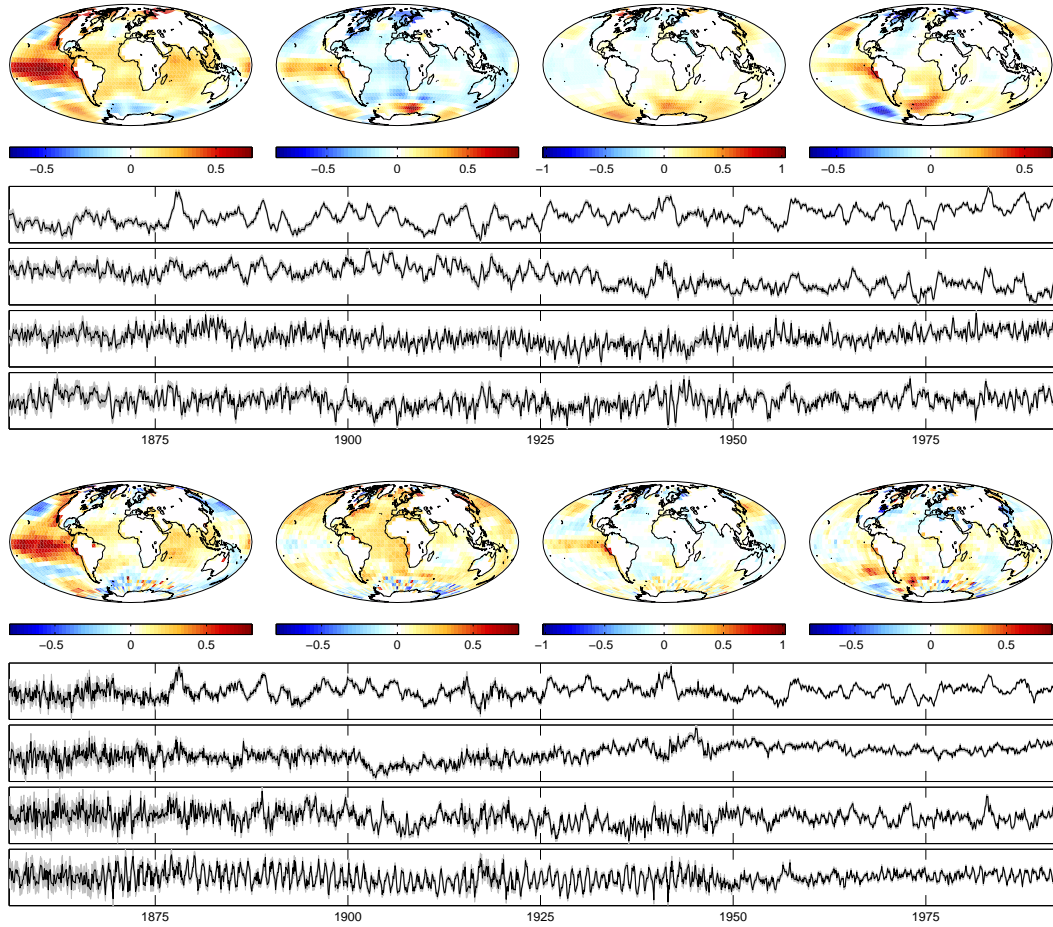

Figure 2: Experimental results for the MOHSST5 dataset. The spatial and temporal patterns of the four most dominating principal components for GPFA (above) and VBPCA (below). The solid lines and gray color in the time series show the mean and two standard deviations of the posterior distribution. The uncertainties of the spatial patterns are not shown, and we saturated the visualizations of the VBPCA spatial components to reduce the effect of the uncertain pole regions.

randomly by sampling from the prior and the weights $\mathbf{W}$ were initialized to zero. The variational EM-algorithm of GPFA was run for 200 iterations. We also applied the variational Bayesian PCA (VBPCA) [2] to the same dataset for comparison. VBPCA was initialized randomly as the initialization did not have much effect on the VBPCA results. Finally, we rotated the GPFA components such that the orthogonal basis in the factor analysis subspace was ordered according to the amount of explained data variance (where the variance was computed by averaging over time). Thus, "GPFA principal components" are mixtures of the original factors found by the algorithm. This was done for comparison with the most prominent patterns found with VBPCA.

Fig. 2 shows the spatial and temporal patterns of the four most dominant principal components for both models. The GPFA principal components and the corresponding spatial patterns are generally smoother, especially in the data-sparse regions, for example, in the period before 1875. The first and the second principal components of GPFA as well as the first and the third components of VBPCA are related to El Niño. We should make a note here that the rotation within the principal subspace may be affected by noise and therefore the components may not be directly comparable. Another observation was that the model efficiently used only some of the 15 slow components: about three very slow and two interannual components had relatively large weights in the loading matrix $\mathbf{W}$. Therefore the selected number of slow components did not affect the results significantly. None

of the periodic components had large weights, which suggests that the fourth VBPCA component might contain artifacts.

Finally, we compared the two models by computing a weighted root mean square reconstruction error on the test set, similarly to [4]. The prediction errors were 0.5714 for GPFA and 0.6180 for VBPCA. The improvement obtained by GPFA can be considered quite significant taking into account the substantial amount of noise in the data.

## 5 Conclusions and discussion

In this work, we proposed a factor analysis model which can be used for modeling spatio-temporal datasets. The model is based on using GP priors for both spatial patterns and time signals corresponding to the hidden factors. The method can be seen as a combination of temporal smoothing, empirical orthogonal functions (EOF) analysis and kriging. The latter two methods are popular in geostatistics (see, e.g., [3]). We presented a learning algorithm that can be applicable to relatively large datasets.

The proposed model was applied to the problem of reconstruction of historical global sea surface temperatures. The current state-of-the-art reconstruction methods [5] are based on the reduced space (i.e. EOF) analysis with smoothness assumptions for the spatial and temporal patterns. That approach is close to probabilistic PCA [14] with fitting a simple auto-regressive model to the posterior means of the hidden factors. Our GPFA model is based on probabilistic formulation of essentially the same modeling assumptions. The gained advantage is that GPFA takes into account the uncertainty about the unknown parameters, it can use all available data and it can combine all modeling assumptions in one estimation procedure. The reconstruction results obtained with GPFA are very promising and they suggest that the proposed model might be able to improve the existing SST reconstructions. The improvement is possible because the method is able to model temporal and spatial phenomena on different scales by using properly selected GPs.

## A The gradients for the updates of GP hyperparameters

The gradient of the first term of (13) w.r.t. a hyperparameter (or inducing input) $\theta$ of any covariance function is given by

$$\frac{1}{2}\operatorname{tr}\left[\left(\mathbf{K}_x^{-1} - \mathbf{A}^{-1}\right)\frac{\partial \mathbf{K}_x}{\partial\theta}\right] - \operatorname{tr}\left[\mathbf{U}\mathbf{K}_{\mathbf{x}x}\mathbf{A}^{-1}\frac{\partial \mathbf{K}_{x\mathbf{x}}}{\partial\theta}\right] + -\frac{1}{2}\mathbf{b}^{\mathsf{T}}\frac{\partial \mathbf{K}_x}{\partial\theta}\mathbf{b} + \mathbf{b}^{\mathsf{T}}\frac{\partial \mathbf{K}_{x\mathbf{x}}}{\partial\theta}(\mathbf{Z}_{:} - \mathbf{U}\mathbf{K}_{\mathbf{x}x}\mathbf{b})$$

where $\mathbf{A} = \mathbf{K}_x + \mathbf{K}_{x\mathbf{x}}\mathbf{U}\mathbf{K}_{\mathbf{x}x}$, $\mathbf{b} = \mathbf{A}^{-1}\mathbf{K}_{x\mathbf{x}}\mathbf{Z}_{:}$. This part is similar to the gradient reported in [12]. Without the sparse approximation, it holds that $\mathbf{K}_x = \mathbf{K}_{\mathbf{x}} = \mathbf{K}_{x\mathbf{x}} = \mathbf{K}_{\mathbf{x}x}$ and the equation simplifies to the regular gradient in GP regression for projected observations $\mathbf{U}^{-1}\mathbf{Z}_{:}$ with the noise covariance $\mathbf{U}^{-1}$. The second part of (13) results in the extra terms

$$\operatorname{tr}\left(\frac{\partial \mathbf{K}_{\mathbf{x}}}{\partial\theta}\mathbf{U}\right) + \operatorname{tr}\left(\frac{\partial \mathbf{K}_x}{\partial\theta}\mathbf{K}_x^{-1}\mathbf{K}_{x\mathbf{x}}\mathbf{U}\mathbf{K}_{\mathbf{x}x}\mathbf{K}_x^{-1}\right) - 2\operatorname{tr}\left(\frac{\partial \mathbf{K}_{\mathbf{x}x}}{\partial\theta}\mathbf{K}_x^{-1}\mathbf{K}_{x\mathbf{x}}\mathbf{U}\right). \qquad (18)$$

The terms in (18) cancel out when the sparse approximation is not used. Both parts of the gradient can be efficiently evaluated using the Cholesky decomposition. The positivity constraints of the hyperparameters can be taken into account by optimizing with respect to the logarithms of the hyperparameters.

**Acknowledgments**

This work was supported in part by the Academy of Finland under the Centers for Excellence in Research program and Alexander Ilin's postdoctoral research project. We would like to thank Alexey Kaplan for fruitful discussions and providing his expertise on the problem of sea surface temperature reconstruction.

**References**

[1] A. Belouchrani, K. A. Meraim, J.-F. Cardoso, and E. Moulines. A blind source separation technique based on second order statistics. *IEEE Transactions on Signal Processing*, 45(2):434–444, 1997.

[2] C. M. Bishop. Variational principal components. In *Proceedings of the 9th International Conference on Artificial Neural Networks (ICANN'99)*, pages 509–514, 1999.

[3] N. Cressie. *Statistics for Spatial Data*. Wiley-Interscience, New York, 1993.

[4] A. Ilin and A. Kaplan. Bayesian PCA for reconstruction of historical sea surface temperatures. In *Proceedings of the International Joint Conference on Neural Networks (IJCNN 2009)*, pages 1322–1327, Atlanta, USA, June 2009.

[5] A. Kaplan, M. Cane, Y. Kushnir, A. Clement, M. Blumenthal, and B. Rajagopalan. Analysis of global sea surface temperatures 1856–1991. *Journal of Geophysical Research*, 103:18567–18589, 1998.

[6] D. E. Parker, P. D. Jones, C. K. Folland, and A. Bevan. Interdecadal changes of surface temperature since the late nineteenth century. *Journal of Geophysical Research*, 99:14373–14399, 1994.

[7] J. Quiñonero-Candela and C. E. Rasmussen. A unifying view of sparse approximate Gaussian process regression. *Journal of Machine Learning Research*, 6:1939–1959, Dec. 2005.

[8] C. E. Rasmussen and C. K. I. Williams. *Gaussian Processes for Machine Learning*. MIT Press, 2006.

[9] J. Särelä and H. Valpola. Denoising source separation. *Journal of Machine Learning Research*, 6:233–272, 2005.

[10] M. N. Schmidt. Function factorization using warped Gaussian processes. In L. Bottou and M. Littman, editors, *Proceedings of the 26th International Conference on Machine Learning (ICML'09)*, pages 921–928, Montreal, June 2009. Omnipress.

[11] M. N. Schmidt and H. Laurberg. Nonnegative matrix factorization with Gaussian process priors. *Computational Intelligence and Neuroscience*, 2008:1–10, 2008.

[12] M. Seeger, C. K. I. Williams, and N. D. Lawrence. Fast forward selection to speed up sparse Gaussian process regression. In *Proceedings of the 9th International Workshop on Artificial Intelligence and Statistics (AISTATS'03)*, pages 205–213, 2003.

[13] Y. W. Teh, M. Seeger, and M. I. Jordan. Semiparametric latent factor models. In *Proceedings of the 10th International Workshop on Artificial Intelligence and Statistics (AISTATS'05)*, pages 333–340, 2005.

[14] M. E. Tipping and C. M. Bishop. Probabilistic principal component analysis. *Journal of the Royal Statistical Society Series B*, 61(3):611–622, 1999.

[15] M. K. Titsias. Variational learning of inducing variables in sparse Gaussian processes. In *Proceedings of the 12th International Workshop on Artificial Intelligence and Statistics (AISTATS'09)*, pages 567–574, 2009.

[16] B. M. Yu, J. P. Cunningham, G. Santhanam, S. I. Ryu, K. V. Shenoy, and M. Sahani. Gaussian-process factor analysis for low-dimensional single-trial analysis of neural population activity. In *Advances in Neural Information Processing Systems 21*, pages 1881–1888. 2009.

## Footnotes

[1]In practical applications, it may be desirable to diminish the effect of uneven sampling over space or time by, for example, using proper weights for different data points.
